# Time Trials on Second-Order and Variable-Learning-Rate Algorithms

**Richard Rohwer**
Centre for Speech Technology Research
Edinburgh University
80, South Bridge
Edinburgh EH1 1HN, SCOTLAND

## Abstract

The performance of seven minimization algorithms are compared on five neural network problems. These include a variable-step-size algorithm, conjugate gradient, and several methods with explicit analytic or numerical approximations to the Hessian.

## 1 Introduction

There are several minimization algorithms in use which in the $n^{th}$ iteration vary the $i^{th}$ coordinate $x_i$ in the direction

$$s_i^{n+1} = r_i^n s_i^n + h_i^n \nabla_i^n \tag{1}$$

where $\nabla_i^n = \frac{dE}{dx_i}\Big|_{x^n}$ is the $i^{th}$ component of the gradient of the error measure $E$ at $x^n$, $s^0 = \nabla^0$, and $r^n$ and $h^n$ are chosen differently in different algorithms. Algorithms also use various methods for choosing the step size $\eta^n$ to be taken along direction $s^n$. In this study, 7 algorithms were compared on a suite of 5 neural network problems. These algorithms are defined in table 1.

### 1.1 The algorithms

The algorithms investigated are Silva and Almeida's variable-step-size algorithm (Silva, 1990) which closely resembles Toolenaere's "SuperSAB" algorithm (Toole-

Table 1: The minimization methods studied.

**Iteration rule:** $\quad z_i^{n+1} = \eta_i^{n+1} s_i^{n+1} \qquad s_i^{n+1} = r_i^n s_i^n + h_i^n \nabla_i^n$

| Method | $r_i^n$ | $h_i^n$ | $\eta^{n+1}$ | other | Parameter settings |
|---|---|---|---|---|---|
| Silva/Toolenaere | $\alpha \gamma_i^n$ | $\gamma_i^n h_i^{n-1}$ | $1,\ E^{n+1} < E^n$ <br> $0,\ E^{n+1} \geq E^n$ | $\gamma_i^n = \begin{cases} u, & \nabla_i^n \cdot \nabla_i^{n-1} > 0 \\ 1, & \nabla_i^n \cdot \nabla_i^{n-1} = 0 \\ d, & \nabla_i^n \cdot \nabla_i^{n-1} < 0 \end{cases}$ | $u = 1.2$ <br> $d = 0.7$ <br> $\alpha = 0.9$ <br> $h_i^0 = 0.5$ <br> $\beta_i^0 = 0$ |
| Conjugate Gradient | $\dfrac{(\nabla^n - \nabla^{n-1}) \cdot \nabla^n}{\nabla^{n-1} \cdot \nabla^{n-1}}$ | $1$ | linesearch | | |
| Analytic (signed) | $0$ | $\dfrac{1}{B(\beta_i^n, \epsilon)}$ | linesearch | $\beta_i^n = cD_i^n + (1-c)\beta_i^{n-1}$ | $c = 0.2$ <br> $\beta_i^0 = 1.0$ <br> $\epsilon = 10^{-14}$ |
| Analytic (abs) | $0$ | $\dfrac{1}{B(\beta_i^n, \epsilon)}$ | linesearch | $\beta_i^n = c|D_i^n| + (1-c)\beta_i^{n-1}$ | $c = 0.2$ <br> $\beta_i^0 = 1.0$ <br> $\epsilon = 10^{-14}$ |
| Analytic (signed; 1-step) | $0$ | $\dfrac{1}{B(\beta_i^n, \epsilon)}$ | $\dfrac{2a|\nabla^n|/|s^{n+1}|}{\sum_j D_j^n D_j^n \hat{D}_j^n}$ | $\beta_i^n = cD_i^n + (1-c)\beta_i^{n-1}$ | $c = 0.2$ <br> $\beta_i^0 = 1.0$ <br> $a = 0.1$ <br> $\epsilon = 10^{-14}$ |
| Analytic (abs, 1-step) | $0$ | $\dfrac{1}{B(\beta_i^n, \epsilon)}$ | $\dfrac{2a|\nabla^n|/|s^{n+1}|}{\sum_j D_j^n D_j^n \hat{D}_j^n}$ | $\beta_i^n = c|D_i^n| + (1-c)\beta_i^{n-1}$ | $c = 0.2$ <br> $\beta_i^0 = 1.0$ <br> $a = 0.1$ <br> $\epsilon = 10^{-14}$ |
| Max (abs) of analytic & numerical | $0$ | $\dfrac{1}{B(\beta_i^n, \epsilon)}$ | linesearch | $\beta_i^n = c\max\left(|D_i^n|, \left|\dfrac{\nabla_i^n - \nabla_i^{n-1}}{B(s_i^n - s_i^{n-1}, \epsilon)}\right|\right)$ $+ (1-c)\beta_i^{n-1}$ | $c = 0.2$ <br> $\beta_i^0 = 1.0$ <br> $a = 0.1$ <br> $\epsilon = 10^{-14}$ |

**Definitions:** $\quad \nabla_i = \dfrac{dE}{dx_i} \qquad D_i = \dfrac{d^2E}{dx_i^2} \qquad \hat{D} = \dfrac{D}{|D|} \qquad B(z, \epsilon) = \begin{cases} z & |z| > \epsilon \\ \epsilon & |z| \leq \epsilon \end{cases} \qquad E^n = E(z^n)$

naere, 1990), conjugate gradient (Press, 1988), and 5 variants of an algorithm advocated by LeCun (LeCun, 1989), which employs an analytic calculation of the diagonal terms of the matrix of second derivatives. (Algorithms involving an approximation of the full Hessian, the inverse of the matrix of second derivatives, were studied by Watrous (Watrous, 1987).) In 4 of these methods the gradient is divided component-wise by a decaying average of either the second derivatives or their absolute values. Dividing by the absolute values assures that $s \cdot \nabla < 0$, and reflects the philosophy that directions with high curvature, be it positive or negative, are

not good ones to follow because the quadratic approximation is likely to break down at short distances. In the remaining method, sketched in (Rohwer, 1990a,b), the gradient is divided componentwise by the maximum of the absolute values of an analytic and numerical calculation of the second derivatives. Again the philosopy is that curvature is to be avoided. The numerical calculation may detect evidence of nearby high curvature at a point where the analytic calculation finds low curvature.

Some algorithms conventionally use a multi-step 1-dimensional "linesearch" to determine how far to proceed in direction $s$, whereas others take a single step according to some formula. A linesearch guarantees descent (more precisely, non-ascent), which is beneficial if local minima pose no threat. Table ?? shows the step-size methods used in this study; the decisions are rather arbitrary. The theoretical basis of the conjugate gradient method is lost if exact linesearches are not used, but it is lost anyway on any non-quadratic function. Silva and Toolenaere's use a single-step method which guarantees descent by retracting any step which does not produce ascent. The method is not a linesearch however, because the step following a retracted step will be in a different direction. Space limitations prohibit a detailed specification of the of the linesearch algorithm and the convergence criteria used. These details may be very important. A longer paper is planned in which they are to be specified, and their influence on performance studied.

## 1.2  The test problems

Two types of problems are used in these tests. One is a strictly-layered 3-layer back propagation network in which the minimization variables are the weights. The test problems are 4-bit parity using 4 hidden nodes, auto-association of 10-bit random patterns using 7 hidden nodes, and the Peterson and Barney vowel classification problem (Peterson, 1952), which uses 2 inputs, 10 hidden nodes, and 10 target nodes. The other type is a fully connected recurrent network trained by the Moving Targets method (Rohwer, 1990a,b). In this case the minimization variables are the weights and the moving targets, which can be regarded as variable training data for the hidden nodes. The limit cycle switching problem and the 100-step context sensitivity problem from these references are the test problems used. In the limit-cycle switching problem, a single target node is required to regularly generate pulses of width proportional to a 2-bit binary number indicated by 2 input nodes. In the 100-step context problem, the training data always has an input pulse at time step 100, and sometimes has an input pulse at time 0. The target node is required to turn on at time 100 if and only if there was an input pulse at time 0.

Each method is tested on each problem with 10 different random initial conditions, except for the parity problem which was done with 100 different initial conditions.

## 1.3  Unconventional nonlinearity

An unconventional form of nonlinearity was used in these tests. The usual $f(x) = 1/(1 + e^{-x})$ presents difficulties when $x \to \pm\infty$ because its derivative becomes very small. This makes the system learn slowly if activations become large. Also, numerical noise becomes serious if expressions such as $f(x)(1 - f(x))$ are used in the derivative calculations. Various cutoff schemes are sometimes used to prevent these problems, but these introduce discontinuities and/or incorrect derivative

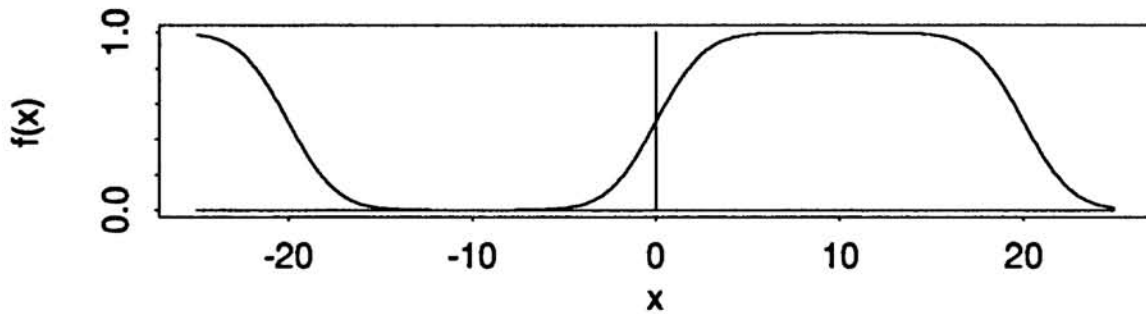

Figure 1: The nonlinearity used.

calculations which present further problems for second-derivative methods. In early work it was found that algorithm performance was highly sensitive to cutoff value (More systematic work on this subject is wanting.), so an entirely different nonlinearity was introduced which is bounded but has reasonably large derivatives for most arguments. This combination of properties can only be had with an oscillatory function. It was also desired to retain the property of $1/(1 + e^{-x})$ that it has large "saturated regions" in which it is approximately constant. The function used is

$$f(x) = \frac{1}{2} + \frac{1}{2(1+\beta)}(1 + \beta \sin(\frac{\pi x}{2\alpha})^2) \sin(\frac{\pi}{2} \sin(\frac{\pi}{2} \sin(\frac{\pi x}{2\alpha}))) \qquad (2)$$

with $\alpha = 10$ and $\beta = 0.02$. This function is graphed in figure 1.

## 2   Results

An algorithm is useful if it produces good solutions quickly. The data for each algorithm-problem pair is divided into separate sets for successful and unsuccessful runs. Success is defined rather arbitrarily as less than 1% error on any target node for all training data in the backpropagation problems. In the Moving Target problems, it is defined in terms of the maximum error on any target node in the freely-running network, the threshold being 5% for the 4-limit-cycle problem and 10% for the 100-step-context problem.

The speed data, measured in number of gradient evaluations, is presented in figure 2, which contains 4 tables, one for each problem except random autoassociation. A maximum of 10000 evaluations was allowed. Each table is divided into 7 columns, one for each algorithm. From left to right, the algorithms are Rohwer's algorithm (max_abs), conjugate gradient (cg), division by unsigned (an_abs) or signed (an_sgn) analytically computed second derivatives and using a linesearch, these two with the linesearch replaced by a single variably-sized step (an_abs_ss and an_sgn_ss) and Silva's algorithm (silva_ss). The data in each of these 7 columns is divided into 3 subcolumns, the first (a) shows all data points, the second (s) shows data for successful runs only, and the third (f) shows data for the failures. Each error bar shows the mean and standard deviation of the data in its column. The all-important little boxes at the base of each column show the proportions of runs in that column's category.

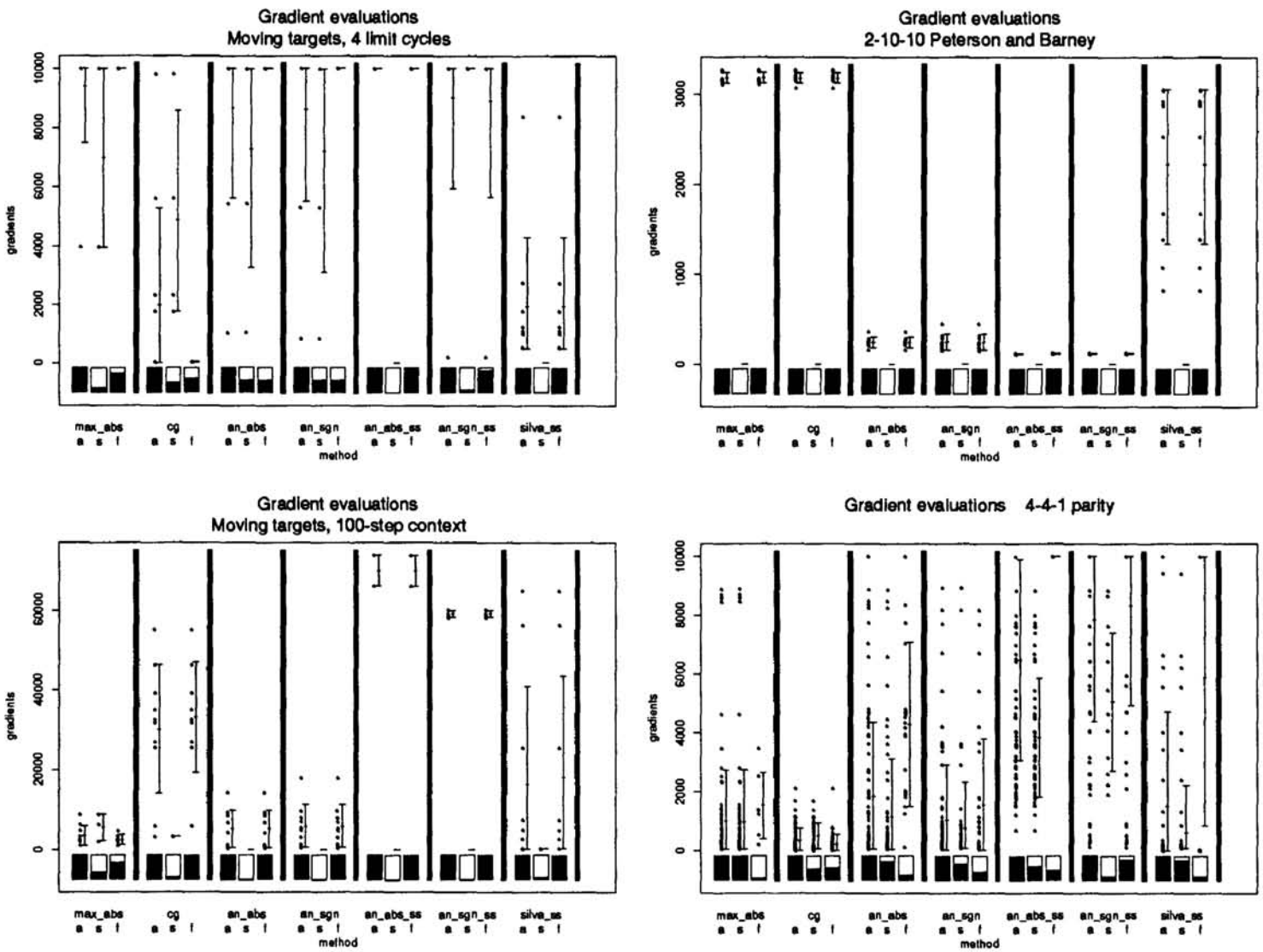

Figure 2: Gradient computations.

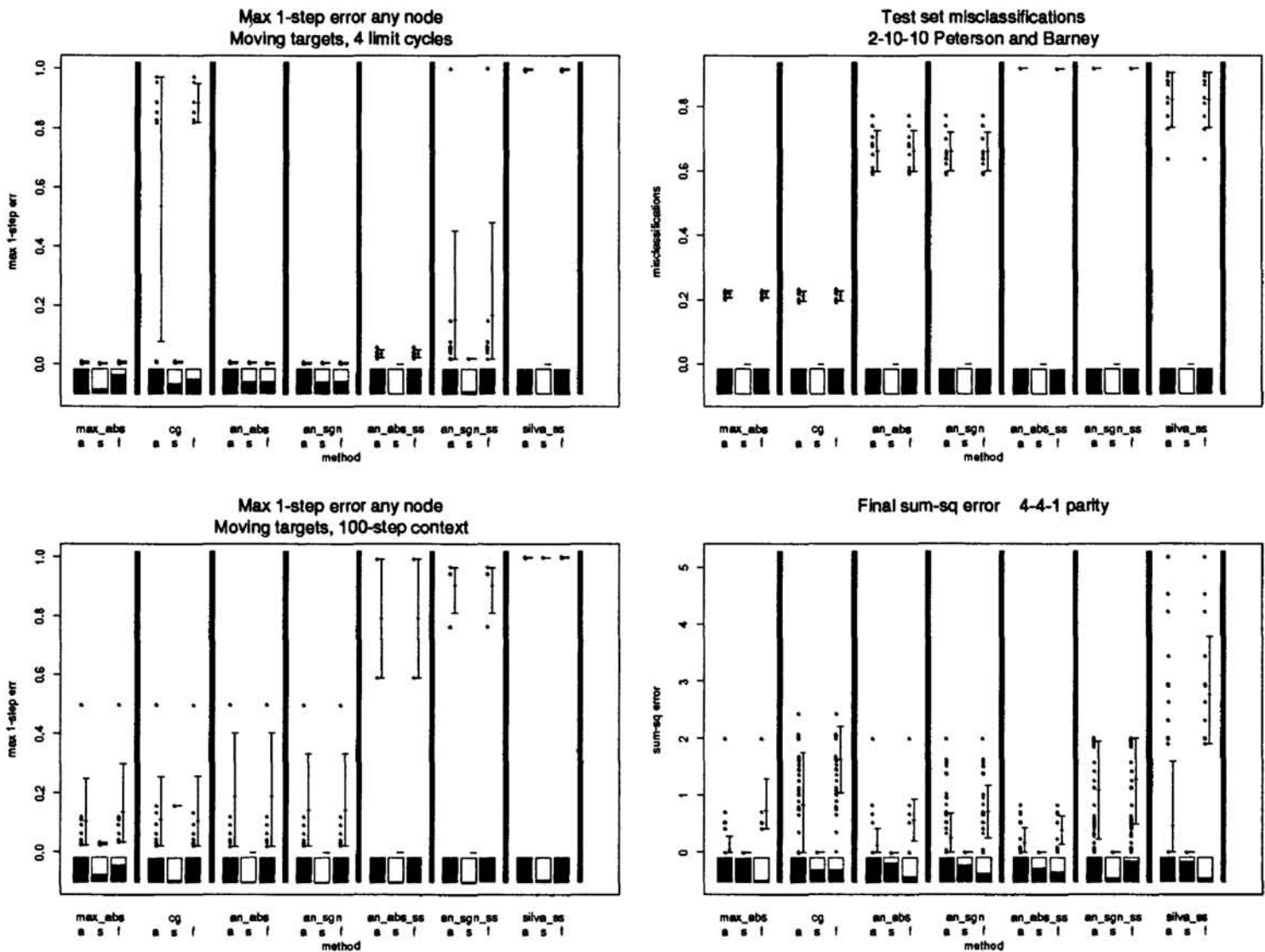

Figure 3: Network performance at error minimum.

The success criteria are quite arbitrary and innapropriate in many cases, so more detailed information on the quality of the solutions is given in Table 3. The maximum error on any target node after one time step, given the moving target values on the previous time step is shown for the Moving Target problems. Test set misclassifications are shown for the Peterson and Barney data, and final sum-squared error is shown for the parity problem.

The random autoassociation results are omitted here to save space. They qualitatively resemble the Peterson and Barney results.

Firm conclusions cannot be drawn, but the linesearch-based algorithms tend to outperform the others. Of these, the conjugate gradient algorithm and Rohwer's algorithm (Rohwer 1990a,b) are usually best.

In recent correspondence with the author, Silva has suggested small changes in his algorithm. In particular, when the algorithm fails to find descent for 5 consecutive iterations, all the learning-rate parameters are halved. Preliminary tests suggest that this change may bring enormous improvements.

## Acknowledgements

This work was supported in part by ESPRIT Basic Research Action 3207 ACTS.

## References

Y. LeCun, et. al. (1989) Generalization and network design strategies. In R. Pfeifer, (ed.), *Connectionism in Perspective*, 143–155. Amsterdam: North Holland.

G. E. Peterson & H. L. Barney. (1952) Control methods used in a study of vowels. *J. Acoustical Soc. of America* 24:175–184.

W. H. Press, et. al. (1988) *Numerical Recipes in C: The Art of Scientific Computing.* Cambridge: Cambridge U. Press

R. Rohwer. (1990a) The 'moving targets' training algorithm. In L. B. Almeida & C. J. Wellekens (eds), *Neural Networks*, Lecture Notes in Computer Science 412:100–109. Berlin: Springer-Verlag.

R. Rohwer. (1990b) The 'moving targets' training algorithm. In D. S. Touretzky (ed.), *Advances in Neural Information Processing Systems* 2:558–565. San Mateo CA: Morgan Kaufmann.

F. M. Silva & L. B. Almeida. (1990). Acceleration techniques for the backpropagation algorithm. In L. B. Almeida & C. J. Wellekens (eds), *Neural Networks*, Lecture Notes in Computer Science 412:110-119. Berlin: Springer-Verlag.

T. Toolenaere. (1990) SuperSAB: Fast Adaptive Back Propagation with Good Scaling Properties. *Neural Networks* 3(5):561–574.

R. Watrous. (1987) Learning algorithms for connectionist networks: Applied gradient methods of nonlinear optimization. In Caudill & Butler (eds.), *IEEE Intl. Conf. on Neural Networks*, II:619–627. San Diego: IEEE.
